# Cyclizing Clusters via Zeta Function of a Graph

**Deli Zhao and Xiaoou Tang**

Department of Information Engineering, Chinese University of Hong Kong
Hong Kong, China
{dlzhao,xtang}@ie.cuhk.edu.hk

## Abstract

Detecting underlying clusters from large-scale data plays a central role in machine learning research. In this paper, we tackle the problem of clustering complex data of multiple distributions and multiple scales. To this end, we develop an algorithm named Zeta $l$-links (Zell) which consists of two parts: Zeta merging with a similarity graph and an initial set of small clusters derived from local $l$-links of samples. More specifically, we propose to structurize a cluster using cycles in the associated subgraph. A new mathematical tool, Zeta function of a graph, is introduced for the integration of all cycles, leading to a structural descriptor of a cluster in determinantal form. The popularity character of a cluster is conceptualized as the global fusion of variations of such a structural descriptor by means of the leave-one-out strategy in the cluster. Zeta merging proceeds, in the hierarchical agglomerative fashion, according to the maximum incremental popularity among all pairwise clusters. Experiments on toy data clustering, imagery pattern clustering, and image segmentation show the competitive performance of Zell. The $98.1\%$ accuracy, in the sense of the normalized mutual information (NMI), is obtained on the FRGC face data of 16028 samples and 466 facial clusters.

## 1  Introduction

Pattern clustering is a classic topic in pattern recognition and machine learning. In general, algorithms for clustering fall into two categories: partitional clustering and hierarchical clustering. Hierarchical clustering proceeds by merging small clusters (agglomerative) or dividing large clusters into small ones (divisive). The key point of agglomerative merging is the measurement of structural affinity between clusters. This paper is devoted to handle the problem of data clustering via hierarchical agglomerative merging.

### 1.1  Related work

The representative algorithms for partitional clustering are the traditional K-means and the latest Affinity Propagation (AP) [1]. It is known that the K-means is sensitive to the selection of initial $K$ centroids. The AP algorithm addresses this issue by that each sample is initially viewed as an examplar and then examplar-to-member and member-to-examplar messages competitively transmit among all samples until a group of good examplars and their corresponding clusters emerge. Besides the superiority of finding good clusters, AP exhibits the surprising ability of handling large-scale data. However, AP is computationally expensive to acquire clusters when the number of clusters is set in advance. Both K-means and AP encounter difficulty on multiple manifolds mixed data.

The classic algorithms for agglomerative clustering include three kinds of linkage algorithms: the single, complete, and average Linkages. Linkages are free from the restriction on data distributions, but are quite sensitive to local noisy links. A novel agglomerative clustering algorithm was recently developed by Ma *et al.* [2] with the lossy coding theory of multivariate mixed data. The core of their algorithm is to characterize the structures of clusters by means of the variational coding length of coding arbitrary two merged clusters against only coding them individually. The coding length

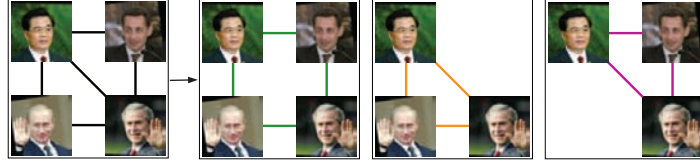

Figure 1: A small graph with four vertices and five edges can be decomposed into three cycles. The complexity of the graph can be characterized by the collective dynamics of these basic cycles.

based algorithm exhibits the exceptional performance for clustering multivariate Gaussian data or subspace data. However, it is not suitable for manifold-valued data.

Spectral clustering algorithms are another group of popular algorithms developed in recent years. The Normalized Cuts (Ncuts) algorithm [3] was developed for image segmentation and data clustering. Ng *et al.*'s algorithm [4] is mainly for data clustering, and Newman's work [5] is applied for community detection in complex networks. Spectral clustering can handle complex data of multiple distributions. However, it is sensitive to noise and the variation of local data scales.

In general, the following four factors pertaining to data are still problematic for most clustering algorithms: 1) mixing distributions such as multivariate Gaussians of different derivations, subspaces of different dimensions, or globally curved manifolds of different dimensions; 2) multiple scales; 3) global sampling densities; and 4) noise. To attack these problems, it is worthwhile to develop new approaches that are conceptually different from existing ones.

## 1.2 Our work

To address issues for complex data clustering, we develop a new clustering approach called Zeta *l*-links, or Zell. The core of the algorithm is based on a new cluster descriptor that is essentially the integration of all cycles in the cluster by means of the Zeta function of the corresponding graph. The Zeta function leads to a rational form of cyclic interactions of members in the cluster, where cycles are employed as primitive structures of clusters. With the cluster descriptor, the popularity of a cluster is quantified as the global fusion of variations of the structural descriptor by the leave-one-out strategy in the cluster. This definition of the popularity is expressible by diagonals of matrix inverse. The structural inference between clusters may be performed with this *popularity character*. Based on the novel popularity character, we propose a clustering method, named *Zeta merging* in the hierarchical agglomerative fashion. This method has no additional assumptions on data distributions and data scales. As a subsidiary procedure for Zeta merging, we present a simple method called *l-links*, to find the initial set of clusters as the input of Zeta merging. The Zell algorithm is the combination of Zeta merging and *l*-links. Directed graph construction is derived from *l*-links.

## 2 Cyclizing a cluster with Zeta function

Our ideas are mainly inspired by recent progress on study of collective dynamics of complex networks. Experiments have validated that the stochastic states of a neuronal network is partially modulated by the information that *cyclically* transmits [6], and that proportions of cycles in a network is strongly relevant to the level of its complexity [7]. Recent studies [8], [9] unveil that short cycles and Hamilton cycles in graphs play a critical role in the structural connectivity and community of a network. These progress inspires us to formalize the structural complexity of a cluster by means of cyclic interactions of its members. As illustrated in Figure 1, the relationship between samples can be characterized by the combination of all cycles in the graph. Thus the structural complexity of the graph can be conveyed by the collective dynamics of these basic cycles. Therefore, we may characterize a cluster with the global combination of structural cycles in the associated graph. To do so, we need to model cycles of different lengths and combine them together as a structural descriptor.

### 2.1 Modeling cycles of equal length

We here model cycles using the sum-product codes to structurize a cluster. Formally, let $\mathcal{C} = \{\mathbf{x}_1, \ldots, \mathbf{x}_n\}$ denote the set of sample vectors in a cluster $\mathcal{C}$. Suppose that $\mathbf{W}$ is the weighted adjacency matrix of the graph associated with $\mathcal{C}$. A vertex of the graph represents a member in $\mathcal{C}$. For generality, the graph is assumed to be directed, meaning that $\mathbf{W}$ may be asymmetric. Let $\gamma_\ell = \{p_1 \to p_2 \to \cdots \to p_{\ell-1} \to p_\ell, p_\ell \to p_1\}$ denote any cycle $\gamma_\ell$ of length $\ell$ defined on $\mathbf{W}$. We apply the factorial codes to retrieve the structural information of cycle $\gamma_\ell$, thus defining $\nu_{\gamma_\ell} = \mathbf{W}_{p_\ell \to p_1} \prod_{k=1}^{\ell-1} \mathbf{W}_{p_k \to p_{k+1}}$, where $\mathbf{W}_{p_k \to p_{k+1}}$ is the $(p_k, p_{k+1})$ entry of $\mathbf{W}$. The value $\nu_{\gamma_\ell}$

provides a kind of degree measure of interactions among $\gamma_\ell$-associated vertices. For the set $\mathcal{K}_\ell$ of all cycles of length $\ell$, the sum-product code $\nu_\ell$ is written as:

$$\nu_\ell = \sum_{\gamma_\ell \in \mathcal{K}_\ell} \nu_{\gamma_\ell} = \sum_{\gamma_\ell \in \mathcal{K}_\ell} \mathbf{W}_{p_\ell \to p_1} \prod_{k=1}^{\ell-1} \mathbf{W}_{p_k \to p_{k+1}}. \tag{1}$$

The value $\nu_\ell$ may be viewed as the quantified indication of global interactions among $\mathcal{C}$ in the $\ell$-cycle scale. The structural complexity of the graph is measured by these quantities of cycles of all different lengths, i.e., $\{\nu_1, \ldots, \nu_\ell, \ldots, \nu_\infty\}$. Further, we need to perform the functional integration of these individual measures. The Zeta function of a graph may play a role for such a task.

## 2.2 Integrating cycles using Zeta function

Zeta functions are widely applied in pure mathematics as tools of performing statistics in number theory, computing algebraic invariants in algebraic geometry, measuring complexities in dynamic systems. The forms of Zeta functions are diverse. The Zeta function we use here is defined as:

$$\zeta_z = \exp\left(\sum_{\ell=1}^{\infty} \nu_\ell \frac{z^\ell}{\ell}\right), \tag{2}$$

where $z$ is a real-valued variable. Here $\zeta_z$ may be viewed as a kind of functional organization of all cycles in $\{\mathcal{K}_1, \ldots, \mathcal{K}_\ell, \ldots, \mathcal{K}_\infty\}$ in a global sense. What's interesting is that $\zeta_z$ admits a rational form [10], which makes the intractable manipulations arising in (1) tractable.

**Theorem 1.** $\zeta_z = 1/\det(\boldsymbol{I} - z\boldsymbol{W})$, where $z < \rho(\boldsymbol{W})$ and $\rho(\boldsymbol{W})$ denotes the spectral radius of the matrix $\boldsymbol{W}$.

From Theorem 1, we see that the global interaction of elements in $\mathcal{C}$ is quantified by a quite simple expression of determinantal form.

## 2.3 Modeling popularity

The popularity of a group of samples means how much these samples in the group is perceived to be a whole cluster. To model the popularity, we need to formalize the complexity descriptor of the cluster $\mathcal{C}$. With the cyclic integration $\zeta_z$ in the preceding section, the complexity of the cluster can be measured by the polynomial entropy $\varepsilon_{\mathcal{C}}$ of logarithm form:

$$\varepsilon_{\mathcal{C}} = \ln \zeta_z = \sum_{\ell=1}^{\infty} \nu_\ell \frac{z^\ell}{\ell} = -\ln \det(\mathbf{I} - z\mathbf{W}). \tag{3}$$

The entropy $\varepsilon_{\mathcal{C}}$ will be employed to model the popularity of $\mathcal{C}$. As we analyze at the beginning of Section 2, cycles are strongly associated with structural communities of a network. To model the popularity, therefore, we may investigate the variational information of cycles by successively leaving one member in $\mathcal{C}$ out. More clearly, let $\chi_{\mathcal{C}}$ denote the popularity character of $\mathcal{C}$. Then $\chi_{\mathcal{C}}$ is defined as the averaged sum of the reductive entropies:

$$\chi_{\mathcal{C}} = \frac{1}{n} \sum_{p=1}^{n} \left(\varepsilon_{\mathcal{C}} - \varepsilon_{\mathcal{C} \backslash \mathbf{x}_p}\right) = \varepsilon_{\mathcal{C}} - \frac{1}{n} \sum_{p=1}^{n} \varepsilon_{\mathcal{C} \backslash \mathbf{x}_p}. \tag{4}$$

Let $T$ denote the transpose operator of a matrix and $\mathbf{e}_p$ is the $p$-th standard basis whose $p$-th element is 1 and 0 elsewhere. We have the following theorem.

**Theorem 2.** $\chi_{\mathcal{C}} = \frac{1}{n} \ln \prod_{p=1}^{n} \boldsymbol{e}_p^T (\boldsymbol{I} - z\boldsymbol{W})^{-1} \boldsymbol{e}_p$.

By analysis of inequalities, we may obtain that $\chi_{\mathcal{C}}$ is bounded as $0 < \chi_{\mathcal{C}} \leq (\varepsilon_{\mathcal{C}}/n)$. The popularity measure $\chi_{\mathcal{C}}$ is a structural character of $\mathcal{C}$, which can be exploited to handle problems in learning such as clustering, ranking, and classification.

The computation of $\chi_{\mathcal{C}}$ is involved with that of the inverse of $(\mathbf{I} - z\mathbf{W})$. In general, the complexity of computing $(\mathbf{I} - z\mathbf{W})^{-1}$ is $O(n^3)$. However, $\chi_{\mathcal{C}}$ is only related to the diagonals of $(\mathbf{I} - z\mathbf{W})^{-1}$ instead of a full dense matrix. This unique property leads the computation of $\chi_{\mathcal{C}}$ to the complexity of $O(n^{1.5})$ by a specialized algorithm for computing diagonals of the inverse of a sparse matrix [11].

## 2.4 Structural affinity measurement

Given a set of initial clusters $\mathcal{C}_c = \{\mathcal{C}_1, \ldots, \mathcal{C}_m\}$ and the adjacency matrix $\mathbf{P}$ of the corresponding samples, the affinities between clusters or data groups can be measured via the corresponding popularity character $\chi_{\mathcal{C}}$. Under our framework, an intuitive inference is that the two clusters that share the largest *reciprocal* popularity have the most consistent structures, meaning the two clusters are most relevant from the structural point of view. Formally, for two given data groups $\mathcal{C}_i$ and $\mathcal{C}_j$ from $\mathcal{C}_c$, the criterion of reciprocal popularity may be written as

$$\delta\chi_{\mathcal{C}_i\cup\mathcal{C}_j} = \delta\chi_{\mathcal{C}_i} + \delta\chi_{\mathcal{C}_j} = (\chi_{\mathcal{C}_i|\mathcal{C}_i\cup\mathcal{C}_j} - \chi_{\mathcal{C}_i}) + (\chi_{\mathcal{C}_j|\mathcal{C}_i\cup\mathcal{C}_j} - \chi_{\mathcal{C}_j}), \tag{5}$$

where the conditional popularity $\chi_{\mathcal{C}_i|\mathcal{C}_i\cup\mathcal{C}_j}$ is defined as $\chi_{\mathcal{C}_i|\mathcal{C}_i\cup\mathcal{C}_j} = \frac{1}{|\mathcal{C}_i|}\ln\prod_{\mathbf{x}_p\in\mathcal{C}_i}\mathbf{e}_p^T(\mathbf{I} - z\mathbf{P}_{\mathcal{C}_i\cup\mathcal{C}_j})^{-1}\mathbf{e}_p$ and $\mathbf{P}_{\mathcal{C}_i\cup\mathcal{C}_j}$ is the submatrix of $\mathbf{P}$ corresponding to the samples in $\mathcal{C}_i$ and $\mathcal{C}_j$. The incremental popularity $\delta\chi_{\mathcal{C}_i}$ embodies the information gain of $\mathcal{C}_i$ after being merged with $\mathcal{C}_j$. The larger the value of $\delta\chi_{\mathcal{C}_i\cup\mathcal{C}_j}$ is, the more likely the two data groups $\mathcal{C}_i$ and $\mathcal{C}_j$ are perceived to be one cluster. Therefore, $\delta\chi_{\mathcal{C}_i\cup\mathcal{C}_j}$ may be exploited to measure the structural affinity between two groups of samples from a whole set of samples.

## 3 Zeta merging

We will develop the clustering algorithm using the structural character $\chi_{\mathcal{C}}$. The automatic detection of the number of clusters are also taken into consideration.

### 3.1 Algorithm of Zeta merging

With the criterion of structural affinity in Section 2.4, it is straightforward to write the procedures of clustering in the hierarchical agglomerative way. The algorithm may proceed from the pair $\{\mathcal{C}_i,\mathcal{C}_j\}$ that has the largest incremental popularity $\delta\chi_{\mathcal{C}_i\cup\mathcal{C}_j}$, i.e., $\{\mathcal{C}_i,\mathcal{C}_j\} = \arg\max_{i,j}\delta\chi_{\mathcal{C}_i\cup\mathcal{C}_j}$. We name the method by *Zeta merging*, whose procedures are provided in Algorithm 1. In general, Zeta merging will proceed smoothly if the damping factor $z$ is bounded as $0 < z < \frac{1}{2\|\mathbf{P}\|}$[1].

---

**Algorithm 1** Zeta merging

---

**inputs**: the weighted adjacency matrix $\mathbf{P}$, the $m$ initial clusters $\mathcal{C}_c = \{\mathcal{C}_1,\ldots,\mathcal{C}_m\}$, and the number $m_c$ ($m_c \le m$) of resulting clusters. Set $t = m$.

**while** 1 **do**

  **if** $t = m_c$ **then** break; **end if**

  Search two clusters $\mathcal{C}_i$ and $\mathcal{C}_j$ such that $\{\mathcal{C}_i,\mathcal{C}_j\} = \arg\max_{\{\mathcal{C}_i,\mathcal{C}_j\}\in\mathcal{C}_c}\delta\chi_{\mathcal{C}_i\cup\mathcal{C}_j}$.

  $\mathcal{C}_c \leftarrow \{\mathcal{C}_c \setminus \{\mathcal{C}_i,\mathcal{C}_j\}\} \cup \{\mathcal{C}_i \cup \mathcal{C}_j\}$;   $t \leftarrow t - 1$.

**end while**

---

The merits of Zeta merging are that it is free from the restriction of data distributions and is less affected by the factor of multiple scales in data. Affinity propagation in Zeta merging proceeds on graph according to cyclic associations, requiring no specification on data distributions. Moreover, the popularity character $\chi_{\mathcal{C}}$ of each cluster is obtained from the averaged amount of variational information conveyed by $\varepsilon_{\mathcal{C}}$. Thus the size of a cluster has little influence on the value $\delta\chi_{\mathcal{C}_i\cup\mathcal{C}_j}$. What's most important is that cycles rooted at each point in $\mathcal{C}$ globally interact with all other points. Thus, the global descriptor $\varepsilon_{\mathcal{C}}$ and the popularity character $\chi_{\mathcal{C}}$ are not sensitive to the local data scale at each point, leading to the robustness of Zeta merging against the variation of data scales.

### 3.2 Number of clusters in Zeta merging

In some circumstances, it is needed to automatically detect the number of underlying clusters from given data. This functionality can be reasonably realized in Zeta merging if each cluster corresponds to a diagonal block structure in $\mathbf{P}$, up to some permutations. The principle is that the minimum $\delta\chi_{\mathcal{C}_i\cup\mathcal{C}_j}$ will be zero when a set of separable clusters emerges, behind which is the mathematical principle that inverting a block-diagonal matrix is equivalent to inverting the matrices on the diagonal blocks. In practice, however, the minimum $\delta\chi_{\mathcal{C}_i\cup\mathcal{C}_j}$ has a jumping variation on the stable part of its curve instead of exactly arriving at zero due to the perturbation of the interlinks between clusters. Then the number of clusters corresponds to the step at the jumping point.

## 4 The Zell algorithm

An issue arising in Zeta merging is the determination of the initial set of clusters. Here, we give a method by performing local single Linkages ( message passing by minimum distances). The method of graph construction is also discussed here.

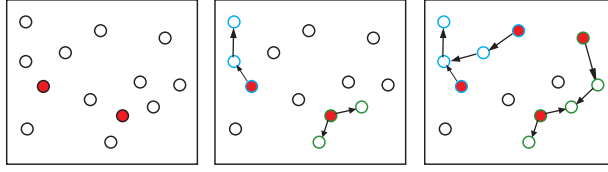

Figure 2: Schematic illustration of $l$-links. From left to right: data with two seed points (red markers), 2-links grown from two seed points, and 2-links from four seed points. The same cluster is denoted by the markers with the same color of edges.

## 4.1 Detecting $l$-links

Given the sample set $\mathcal{C}_y = \{\mathbf{y}_1, \ldots, \mathbf{y}_{m_o}\}$, we first get the set $\mathcal{S}_i^{2K}$ of $2K$ nearest neighbors for the point $\mathbf{y}_i$. Then from $\mathbf{y}_i$, messages are passed among $\mathcal{S}_i^{2K}$ in the sense of minimum distances (or general dissimilarities), thus locally forming an acyclic directed subgraph at each point. We call such an acyclic directed subgraph by $l$-links, where $l$ is the number of steps of message passing among $\mathcal{S}_i^{2K}$. In general, $l$ is a small integer, e.g., $l \in \{2, 3, 4, \ldots\}$. The further manipulation is to merge $l$-links that share common vertices. A simple schematic example is shown in Figure 2. The specific procedures are provided in Algorithm 2.

---

**Algorithm 2** Detecting $l$-links

**inputs**: the sample set $\mathcal{C}_y = \{\mathbf{y}_1, \ldots, \mathbf{y}_{m_o}\}$, the number $l$ of $l$-links, the number $K$ of nearest neighbors for each point, where $l < K$.

Initialization: $\mathcal{C}_c = \{\mathcal{C}_i | \mathcal{C}_i = \{\mathbf{y}_i\}, i = 1, \ldots, m_o\}$ and $q = 1$.

**for** $i$ from 1 to $m_o$ **do**

    Search $2K$ nearest neighbors of $\mathbf{y}_i$ and form $\mathcal{S}_i^{2K}$.

    Iteratively perform $\mathcal{C}_i \leftarrow \mathcal{C}_i \cup \{\mathbf{y}_j\}$ if $\mathbf{y}_j = \arg \min\limits_{\mathbf{y}_j \in \mathcal{S}_i^{2K}} \min\limits_{\mathbf{y} \in \mathcal{C}_i} \text{distance}(\mathbf{y}, \mathbf{y}_j)$, until $|\mathcal{C}_i| \geq l$.

    Perform $\mathcal{C}_j \leftarrow \mathcal{C}_i \cup \mathcal{C}_j$, $\mathcal{C}_c \leftarrow \mathcal{C}_c \setminus \mathcal{C}_i$, and $q \leftarrow q + 1$, if $|\mathcal{C}_i \cap \mathcal{C}_j| > 0$, where $j = 1, \ldots, q$.

**end for**

---

## 4.2 Graph construction

The directional connectivity of $l$-links leads us to build a directed graph whose vertex $\mathbf{y}_i$ directionally points to its $K$ nearest neighbors. The method of graph construction is presented in Algorithm 3. The free parameter $\sigma$ in (6) is estimated according to the criterion that the geometric mean of all similarities between each point and its three nearest neighbors is set to be $a$, where $a$ is a given parameter in $(0, 1]$. It is easy to know that $\rho(\mathbf{P}) < 1$ here.

---

**Algorithm 3** Directed graph construction

**inputs**: the sample set $\mathcal{C}_y$, the number $K$ of nearest neighbors, and a free parameter $a \in (0, 1]$.

Estimate the parameter $\sigma$ by $\sigma^2 = -\frac{1}{m_o \ln a} \sum_{\mathbf{y}_i \in \mathcal{C}_y} \sum_{\mathbf{y}_j \in \mathcal{S}_i^3} [\text{distance}(\mathbf{y}_i, \mathbf{y}_j)]^2$.

Define the entry of the $i$-th row and $j$-th column of the weighted adjacency matrix $\mathbf{P}$ as

$$\mathbf{P}_{i \to j} = \begin{cases} \exp\left(-\frac{[\text{distance}(\mathbf{y}_i, \mathbf{y}_j)]^2}{\sigma^2}\right), & \text{if } \mathbf{y}_j \in \mathcal{S}_i^K, \\ 0, & \text{otherwise.} \end{cases} \quad (6)$$

Perform the sum-to-one operation for each row, i.e., $\mathbf{P}_{i \to j} \leftarrow \mathbf{P}_{i \to j} / \sum_{j=1}^{m_o} \mathbf{P}_{i \to j}$.

---

## 4.3 Zeta $l$-links (Zell)

Our algorithm for data clustering is in effect to perform *Zeta merging* on the initial set of small clusters derived from $l$-links. So, we name our algorithm by *Zeta l-links*, or Zell. The complete implementation of the Zell algorithm is to consecutively perform Algorithm 3, Algorithm 2, and Algorithm 1. In practice , the steps in Algorithm 3 and Algorithm 2 are operated together to enhance

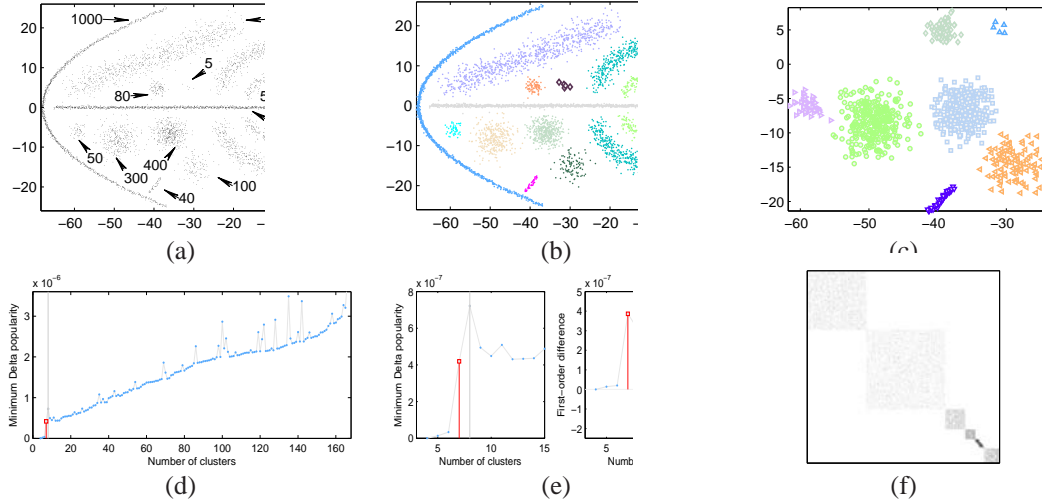

Figure 3: Clustering on toy data. (a) Generated data of 12 clusters. The number of each cluster is shown in the figure. The data are of different distributions, consisting of multiple manifolds (two circles and a hyperbola), subspaces (two pieces of lines and a piece of the rectangular strip), and six Gaussians. The densities of clusters are diverse. The differences between the sizes of different clusters are large. The scales of the data vary. For each cluster in the manifold and subspace data, the points are randomly generated with different deviations. (b) Clusters yielded by Zell (given number of clusters). The different colors denote different clusters. (c) Clusters automatically detected by Zell on the data composed by six Gaussians and the short line. (d) Curve of minimum Delta popularity ($\delta\chi$). (e) Enlarged part of (d) and the curve of its first-order differences. The point marked by the square is the detected jumping point. (f) The block structures of $\mathbf{P}$ corresponding to the data in (c).

the efficiency of Zell. Zeta merging may also be combined with K-means and Affinity Propagation for clustering. These two algorithms work well for producing small clusters. So, they can be employed to generate initial clusters as the input of Zeta merging.

## 5  Experiment

Experiments are conducted on clustering toy data, hand-written digits and cropped faces from captured images, and segmenting images to test the performance of Zell. The quantitative performance of the algorithms is measured by the normalized mutual information (NMI) [12] which is widely used in learning communities. The NMI quantifies the normalized statistical information shared between two distributions. The larger the NMI is, the better the clustering performance of the algorithm is.

Four representative algorithms are taken into comparison, i.e., K-centers, (average) Linkage, Affinity Propagation (AP), and Normalized Cuts (Ncuts). Here we use K-centers instead of K-means because it can handle the case where distances between points are not measured by Euclidean norms. For fair comparison, we run Ncuts on the graph whose parameters are set the same with the graph used by Zell. The parameters for Zell are set as $z = 0.01$, $a = 0.95$, $K = 20$, and $l = 2$.

### 5.1  On toy data

We first perform an experiment on a group of toy data of diverse distributions with multiple densities, multiple scales, and significantly different sizes of clusters. As shown in Figures 3 (b) and (c), the Zell algorithm accurately detects the underlying clusters out. Particularly, Zell is capable of simultaneously differentiating the cluster with five members and the cluster with 1500 members. This functionality is critically *important for finding genes* from microarray expressions in bioinformatics. Figures 3 (d) and (e) show the curves of minimum variational $\delta\chi$ (for the data in Figure 3 (c)) where the number of clusters is determined at the largest gap of the curve in the stable part. However, the method presented in Section 3.2 fails to automatically detect the number of clusters for the data in Figure 3 (a), because the corresponding $\mathbf{P}$ matrix has no clear diagonal block structures.

Table 1: Imagery data. MNIST and USPS: digit databases. ORL and FRGC: face databases. The last row shows the numbers of clusters automatically detected by Zell on the five data sets.

| Data set | MNIST | USPS | ORL | sFRGC | FRGC |
|---|---|---|---|---|---|
| Number of samples | 5139 | 11000 | 400 | 11092 | 16028 |
| Number of clusters | 5 | 10 | 40 | 186 | 466 |
| Average number of each cluster | $1027 \pm 64$ | $1100 \pm 0$ | $10 \pm 0$ | $60 \pm 14$ | $34 \pm 24$ |
| Dimension of each sample | 784 | 256 | 2891 | 2891 | 2891 |
| Detected number of clusters | 11 | 8 | 85 ($K = 5$) | 229 | 511 |

Table 2: Quantitative clustering results on imagery data. NMI: normalized mutual information. The 'pref' means the preference value used in Affinity Propagation for clustering of given numbers. $K = 5$ for the ORL data set.

| Algorithm | | K-centers | Linkage | Ncuts | Affinity propagation (pref) | Zell |
|---|---|---|---|---|---|---|
| NMI | MNIST | 0.228 | 0.496 | 0.737 | 0.451 (-871906470) | **0.865** |
| | USPS | 0.183 | 0.095 | 0.443 | 0.313 (-417749850) | **0.772** |
| | ORL | 0.393 | 0.878 | 0.939 | 0.877 (-6268) | **0.940** |
| | sFRGC | 0.106 | 0.934 | 0.953 | 0.899 (-16050) | *0.988* |
| | FRGC | 0.187 | 0.950 | 0.924 | 0.906 (-7877) | *0.981* |

## 5.2   On imagery data

The imagery patterns we adopt are the hand-written digits in the MNIST and USPS databases and the facial images in the ORL and FRGC (Face Recognition Grand Challenge, http://www.frvt.org/FRGC/) databases. The MNIST and USPS data sets are downloaded from Sam Roweis's homepage (http://www.cs.toronto.edu/~roweis). For MNIST, we select all the images of digits from 0 to 4 in the testing set for experiment. For FRGC, we use the facial images in the target set of experiment 4 in the FRGC version 2. Besides the whole target set, we also select a subset from it. Such persons are selected as another group of clusters if the number of faces for each person is no less than forty. The information of data sets is provided in Table 1. For digit patterns, the Frobenius norm is employed to measure distances of digit pairs without feature extraction. For face patterns, however, we extract visual features of each face by means of the local binary pattern algorithm. The Chi-square metric is exploited to compute distances, defined as distance$(\hat{\mathbf{y}}, \check{\mathbf{y}}) = \sum_i \frac{(\hat{\mathbf{y}}_i - \check{\mathbf{y}}_i)^2}{\hat{\mathbf{y}}_i + \check{\mathbf{y}}_i}$.

The quantitative results are given in Table 2. We see that Zell consistently outperforms the other algorithms across the five data sets. In particular, the performance of Zell is encouraging on the FRGC data set which has the largest numbers of clusters and samples. As reported in [1], AP does significantly outperform K-centers. However, AP shows the unsatisfactory performance on the digit data where the manifold structures may occur due to that the styles of digits vary significantly. The average Linkage also exhibits such phenomena. The results achieved by Ncuts are also competitive. However, Ncuts is overall unstable, for example, yielding the low accuracy on the USPS data. The results in Tabel 3 confirms the stability of Zell over the variations of free parameters. Actually, $l$ affects the performance of Zell when it is larger, because it may incur incorrect initial clusters.

## 5.3   Image segmentation

We show several examples of the application of Zell on image segmentation from the Berkeley segmentation database. The weighted adjacency matrix $\mathbf{P}$ is defined as $\mathbf{P}_{i \to j} = \exp(-\frac{(I_i - I_j)^2}{\sigma^2})$ if $I_j \in \mathcal{N}_i^8$ and 0 otherwise, where $I_i$ is the intensity value of an image and $\mathcal{N}_i^8$ denotes the set of pixels in the 8-neighborhood of $I_i$. Figure 4 displays the segmentation results of different numbers of segments for each image. Overall, attentional regions are merged by Zell. Note the small attentional regions take the priorities of being merged than the large ones. Therefore, Zell yields many small attentional regions as final clusters.

## 6   Conclusion

An algorithm, named Zell, has been developed for data clustering. The cyclization of a cluster is the fundamental principle of Zell. The key point of the algorithm is the integration of structural cycles but Zeta function of a graph. A popularity character of measuring the compactness of the cluster is defined via Zeta function, on which the core of Zell for agglomerative clustering is based. An

Table 3: Results yielded by Zell over variations of free parameters on the sFRGC data. The initial set is $\{z = 0.01, a = 0.95, K = 20, l = 3\}$. When one of them varies, the other keep invariant.

| Parameter | $z$ | $a$ | $K$ | $l$ |
|---|---|---|---|---|
| Range | $10^{-\{1,2,3,4\}}$ | $0.2 \times \{1, 2, 3, 4, 4.75\}$ | $10 \times \{2, 3, 4, 5\}$ | $\{2, 3, 4\}$ |
| NMI | $0.988 \pm 0$ | $0.988 \pm 0.00019$ | $0.987 \pm 0.0015$ | $0.988 \pm 0.0002$ |

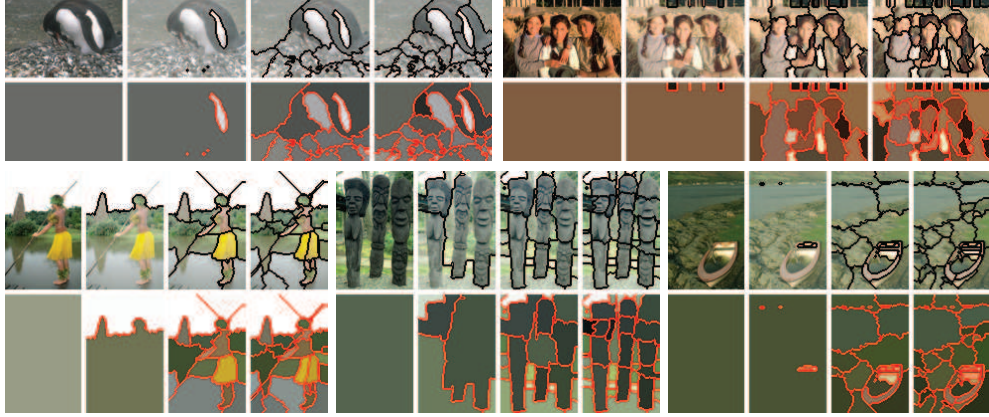

Figure 4: Image segmentation by Zell from the Berkeley segmentation database.

approach for finding initial small clusters is presented, which is based on the merging of local links among samples. The directed graph used in this paper is derived from the directionality of $l$-links. Experimental results on toy data, hand-written digits, facial images, and image segmentation show the competitive performance of Zell. We hope that Zell brings a new perspective on complex data clustering.

# Acknowledgement

We thank Yaokun Wu and Sergey Savchenko for their continuing help on algebraic graph theory. We are also grateful of the interesting discussion with Yi Ma and John Wright on clustering and classification. Feng Li and Xiaodi Hou are acknowledged due to their kind help. The reviewers' insightful comments and suggestions are also greatly appreciated.

## Footnotes

[1]Interested one may refer to the full version of this paper for proofs.

# References

[1] Frey, B.J. & Dueck, D. (2007) Clustering by passing messages between data points. *Science* **315**:972-976.

[2] Ma, Y. Derksen, H. Hong, W. & Wright, J. (2007) Segmentation of multivariate mixed data via lossy data coding and compression. *IEEE Trans. on Pattern Recognition and Machine Intelligence* **29**:1546-1562.

[3] Shi, J.B. & Malik, J. (2000) Normalized cuts and image segmentation. *IEEE Trans. on Pattern Recognition and Machine Intelligence* **22**(8):888-905.

[4] Ng, A.Y., Jordan, M.I. & Weiss, Y. (2001) On spectral clustering: analysis and an algorithm. *Advances in Neural Information Processing Systems*. Cambridge, MA: MIT Press.

[5] Newman, M.E.J. (2006) Finding community structure in networks using the eigenvectors of matrices. *Physical Review E* **74**(3).

[6] Destexhe, A. & Contreras, D. (2006) Neuronal computations with stochastic network states. *Science*, **314**(6):85-90.

[7] Sporns, O. Tononi, G. & Edelman, G.M. (2000) Theoretical neuroanatomy: relating anatomical and functional connectivity in graphs and cortical connection matrices. *Cerebral Cortex*, **10**:127-141.

[8] Bagrow, J. Bollt, E. & Costa, L.F. (2007) On short cycles and their role in network structure. *http://arxiv.org/abs/cond-mat/0612502*.

[9] Bianconi, G. & Marsili, M. (2005) Loops of any size and Hamilton cycles in random scale-free networks. *Journal of Statistical Mechanics*, **P06005**.

[10] Savchenko, S.V. (1993) The zeta-function and Gibbs measures. *Russ. Math. Surv.* **48**(1):189-190.

[11] Li, S. Ahmed, S. Klimeck, G. & Darve, E. (2008) Computing entries of the inverse of a sparse matrix using the FIND algorithm. *Journal of Computational Physics* **227**:9408-9427.

[12] Strehl, A. & Ghosh, J. (2002) Cluster ensembles — a knowledge reuse framework for combining multiple partitions. *Journal of Machine Learning Research* **3**:583617.
